# Auto-Regressive HMM Inference with Incomplete Data for Short-Horizon Wind Forecasting

**Chris Barber**
EE and Computer Science
University of Wisconsin-Milwaukee, USA

**Joseph Bockhorst**
EE and Computer Science
University of Wisconsin-Milwaukee, USA

**Paul Roebber**
Atmospheric Science
University of Wisconsin-Milwaukee, USA

## Abstract

Accurate short-term wind forecasts (STWFs), with time horizons from 0.5 to 6 hours, are essential for efficient integration of wind power to the electrical power grid. Physical models based on numerical weather predictions are currently not competitive, and research on machine learning approaches is ongoing. Two major challenges confronting these efforts are missing observations and weather-regime induced dependency shifts among wind variables. In this paper we introduce approaches that address both of these challenges. We describe a new regime-aware approach to STWF that use auto-regressive hidden Markov models (AR-HMM), a subclass of conditional linear Gaussian (CLG) models. Although AR-HMMs are a natural representation for weather regimes, as with CLG models in general, exact inference is NP-hard when observations are missing (Lerner and Parr, 2001). We introduce a simple approximate inference method for AR-HMMs, which we believe has applications in other problem domains. In an empirical evaluation on publicly available wind data from two geographically distinct regions, our approach makes significantly more accurate predictions than baseline models, and uncovers meteorologically relevant regimes.

## 1 Introduction

Accurate wind speed and direction forecasts are essential for efficient integration of wind energy into electrical transmission systems. The importance of wind forecasts for the wind energy industry stems from three facts: 1) for reliability and safety the aggregate power produced and consumed throughout a power system must be nearly in balance at all times, 2) because it depends strongly on wind speed and direction, the power output of a wind farm is highly variable, and 3) efficient and cost effective energy storage mechanisms do not exist. A recent estimate placed the value of a perfect forecast at \$3Billion annually (Piwko and Jordan, 2010) for the United States power system of 2030 envisioned by the Department of Energy (Lindenberg, 2008). Because information on the 30 minute to six-hour time horizon is actionable for many control decisions, and the current state-of-the-art is considered inadequate, there has been a recent surge of interest in improving forecasts in this range.

The short-term wind forecasting (STWF) problem presents numerous challenges to the modeler. Data produced by the current observation network are sparse relative to the temporal and spatial scale of weather events that drive short-term changes in wind features; observations are frequently missing or corrupted; quality training sets with multiple years of turbine height ($\sim$80 m) wind observations at numerous sites are typically not available; transfer of learned models (Caruana, 1997) across

wind farms is difficult, and because of the dynamic nature of weather the spatial and temporal dependencies of wind features within a geographical region are not fixed.

Numerical weather predictions (NWP) methods are the primary means for producing the large-scale weather forecasts used throughout the world, but are not competitive for STWF. In fact, NWP based wind speed predictions are less accurate than "persistence" forecasts (Giebel, 2003), a surprisingly robust method for time horizons less than a few hours. Approaches to STWF include ARMA models (Marti et al., 2004), support vector machines (Zheng and Kusiak, 2009), and other data mining methods (Kusiak et al., 2009), but with the exception of two methods (Gneiting et al., 2006; Pinson and Madsen, 2008) these do not consider dependency dynamics. Gneiting et al. (2006) "hard code" their regimes based on wind direction, while Pinson and Madsen (2008) learns regimes for a single forecasting site with complete data. At the time of writing we are unaware of any previous STWF work which simultaneously learns regimes, incorporates multiple observation sites, and accepts missing observations.

We propose a novel approach to STWF that automatically reasons about learned weather regimes across multiple sites while naturally handling missing observations. Our approach is based on switching *conditional linear Gaussian* (CLG) models, variously known as switching vector autoregressive models or autoregressive hidden Markov models. For an overview of CLG models, see Koller and Friedman (2009, Chap. 14). Since exact inference in CLG models with incomplete data is NP-hard (Lerner and Parr, 2001), we pursue approximate methods. We introduce a novel and simple approximate inference approach that exploits the tendency of regimes to persist for several hours. Predictions by our learned models are significantly better than baseline *persistence* predictions in experiments on national climatic data center (NCDC) data from two sites in the United States: one in the Pacific Northwest and one in southern Wisconsin. Inspection of the learned models show that our approach learns meteorologically interesting regimes.

Switching CLG models have been applied in other domains where missing observations are an issue, such as meteorology (Tang, 2004; Paroli et al., 2005), epidemiology (Toscani et al., 2010) and econometrics (Perez-Quiros et al., 2010). Some approaches are able to avoid the issue of missing data by throwing out affected timesteps, or by imputing values through a variety of domain-specific techniques. Alternatively, *Markov Chain Monte Carlo* parameter estimation techniques have been applied. Our approach may be an attractive alternative in these domains, offering a solution that does not require deletion or imputation.

## 2   Methods

We consider the setting in which wind observations from a set of $M$ stations arrive at regular intervals (hourly in our experiments). Let $\mathbf{U_t}$ and $\mathbf{V_t}$ be $M$-by-1 vector of random variables for the $u$ and $v$ components of the wind at all sites at time $t$. We use $\mathbf{W_t} = [1\ \mathbf{U_t'}\ \mathbf{V_t'}]'$ to refer to both $\mathbf{U_t}$ and $\mathbf{V_t}$[1], and we denote settings to random variables with lowercase letters, for example $\mathbf{w}_t$.

Our approach to STWF is based on auto-regressive HMMs where at each time $t$ we have a single discrete random variable $R_t$ that represents the active regime, and a continuous valued vector random variable $\mathbf{W}_t$ that represents measured wind speeds. As local probabilities are linear Gaussian (LG) we denote the model in which the regime variables $R_t$ have cardinality $C$ by $\mathsf{AR\text{-}LG}(C)$. Thus, $\mathsf{AR\text{-}LG}(1)$ is a traditional AR model. Figure 1 shows example graphical models.

The local conditional distributions, $\Pr(R_{t+1}|R_t)$ and $\Pr(\mathbf{W}_t|\mathbf{W}_{t-1}, R_t)$, are shared across time. We represent $\Pr(R_{t+1}|R_t)$ by the $C$-by-$C$ transition matrix $T$ where $T(r,s) > 0$ is the probability of transitioning from regime $r$ to regime $s$. Since weather regimes tend to persist for multiple hours, the self-transition probabilities $T(r,r)$ are typically the largest. The local distributions for the continuous variables are linear Gaussian, $\Pr(\mathbf{w}_t|\mathbf{w}_{t-1}, R_t = r) = \mathcal{N}(\mathbf{B}(r)\mathbf{w}_{t-1}, \mathbf{Q}(r))$, where $\mathbf{B}(r)$ is the $2M$-by-$2M$ regression matrix for regime $r$, row $j$ of $\mathbf{B}(r)$ is the regression vector for the $j^{th}$ component of $\mathbf{w}_t$, $\mathbf{Q}(r)$ is the regime's covariance matrix, and $\mathcal{N}(\boldsymbol{\mu}, \Sigma)$ is the multivariate Gaussian (MVG) with mean $\boldsymbol{\mu}$ and covariance $\Sigma$. The joint probability of a setting to all variables

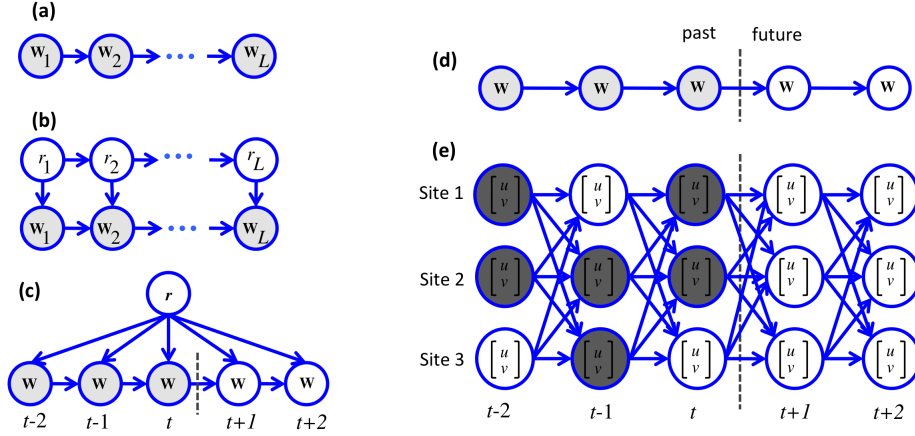

**Figure 1:** Graphical structures of wind speed models. Darkly shaded nodes are observed, lightly shaded nodes are partially observed, and unshaded nodes are unobserved. (a) Auto-regressive linear Gaussian (AR-LG(1)) for data set with $L$ time steps. (b) Auto-regressive HMM (AR-LG($C$), $C > 1$). Exact inference in (b) with missing observations is NP-hard (Lerner and Parr, 2001). (c) Truncated AR-LG($C$) HOMO approximation of (b) for predictions of wind speeds at target time $t + h$ made at $t$ with $K = 2$ and horizon $h = 2$. Our approximation assumes the regime does not change in the window $t - K$ to $t + h$. (d) Truncated (non-conditional) AR-LG(1) analogous to (c). (e) Detailed structure of (d) for 3 sites showing within time-slice conditional independencies and assorted missing observations.

for $L$ time steps is

$$\Pr(r_1, \mathbf{w}_1, \cdots, r_L, \mathbf{w}_L) = \Pr(r_1) \Pr(\mathbf{w}_1|r_1) \prod_{t=2}^{L} \Pr(r_t|r_{t-1}) \Pr(\mathbf{w}_t|r_t, \mathbf{w}_{t-1})$$

$$= \nu(r_1) \mathcal{N}\left(\mathbf{w}_1; \boldsymbol{\mu}_1(r_1), \mathbf{Q}_1(r_1)\right) \prod_{t=2}^{L} T(r_{t-1}, r_t) \mathcal{N}\left(\mathbf{w}_t; \mathbf{B}(r_t)\mathbf{w}_{t-1}, \mathbf{Q}(r_t)\right)$$

where $\nu$ is the initial regime distribution, the observations at $t = 1$ for regime $r$ are Gaussian with mean $\boldsymbol{\mu}_1(r)$ and covariance $\mathbf{Q}_1(r)$, and $\mathcal{N}()$ with three arguments denotes MVG density. We set $\nu$ to the stationary state-distribution, given by the eigenvector of $T'$ associated with eigenvalue 1. We train model parameters with standard EM methods for conditional linear Gaussian (CLG) models (Murphy, 1998), except that the E-step uses approximate inference.

## 2.1  Approximate Inference Methods

Consider a length $L$ time series and let $\mathcal{W} = (\mathbf{W}_1, \cdots, \mathbf{W}_L)$ refer to the continuous variables. We denote the sequence of partial observations as $\dot{\mathbf{w}}_{1:L} = (\dot{\mathbf{w}}_1, \dot{\mathbf{w}}_2, \cdots \dot{\mathbf{w}}_L)$ where the "dot" notation $\dot{\mathbf{w}}_t$ denotes a potentially incomplete vector with missing data. Our inference tasks are to calculate $\Pr(\mathbf{W}_{t-1}, \mathbf{W}_t, R_t|\dot{\mathbf{w}}_{1:L})$ and $\Pr(R_{t-1}, R_t|\dot{\mathbf{w}}_{1:L})$ while training to compute the expected sufficient statistics needed for estimation of CLG parameters using EM (Murphy, 1998), and to compute $\Pr(\mathbf{W}_{t+H}|\dot{\mathbf{w}}_{1:t})$ for horizon $H$ forecasting at time $t$.

In AR-LG(1) models with no discrete variables (Figure 1a), the chain structure permits efficient exact inference using message passing techniques (Weiss and Freeman, 2001). For general AR-LG($C$) models, however, the posterior distributions over unobserved continuous variables are mixtures of exponentially many Gaussians and exact inference is NP-hard (Lerner and Parr, 2001). Specifically, $\Pr(\mathbf{W}_{t+H}|\dot{\mathbf{w}}_{1:t})$ has $C^{d+H}$ component Gaussians, one for each setting of $R_{t-d+1}, \cdots, R_{t+H}$, where $d$ is the number of contiguous time steps in the suffix of $\dot{\mathbf{w}}_{1:t}$ with at least one missing observation. The training posteriors $\Pr(\mathbf{W}_{t-1}, \mathbf{W}_t|\dot{\mathbf{w}}_{1:L})$ have $C^{d_l+2+d_r}$ components where $d_l$ and $d_r$ are the number of consecutive time steps to the left of $t - 1$ and right of $t$ with at least one missing observation. Because of the nature of data collection, most wind data sets with multiple sites will have a number of missing observations. Indeed, our Wisconsin (21 sites) and Pacific Northwest (24 sites) data sets have only 5.6% and 6.4% hours of complete data, respectively. Missing observations

are by no means unique to wind. Lauritzen's approach (Lauritzen and Jensen, 2001) for exact inference in conditional linear Gaussian models offers no relief as the clique sizes in the strongly rooted junction trees are exponentially large for $\mathsf{AR\text{-}LG}(C)$ models.

Since exact inference is intractable we investigate approximate methods. We first make a simplification that involves focusing only on observations temporally close to the inference variables. We ignore observations more than $K$ time-steps from $t$. For example, we approximate $\Pr(\mathbf{W}_{t-1}, \mathbf{W}_t | \dot{\mathbf{w}}_{1:L})$ by a truncated model $\Pr(\mathbf{W}_{t-1}, \mathbf{W}_t | \dot{\mathbf{w}}_{t-K:t+K})$. While inference in the truncated model will be less costly than in the full model, it is still $O(C^{2K+1})$ in the worst case, which is prohibitive for moderate $K$ on large datasets.

Our approaches are based on the general concept of *pruning* (Koller and Friedman, 2009, Chap. 14), where all but $n$ mixture components are discarded, in order to approximate a posterior distribution with a prohibitive number of components. Let $P(V)$ refer to a desired posterior distribution under the truncated model given evidence, which we assume has at least one missing observation at each time step. $P(V)$ is a mixture of Gaussians with an exponential number of components $N$, which we write $P(V) = \sum_{j=1}^{N} \omega_j p_j(V)$. Each mixing proportion $\omega_j$ is associated with a regime state sequence and $p_j(V)$ is the posterior Gaussian for that sequence. We approximate $P(V)$ by the distribution $Q(V) = \sum_{j=1}^{n} \pi_j q_j(V)$ with a much smaller number components $n$ in which each component in $Q$ is equal to one component in $P$. Without loss of generality we re-order components of $P$ so that the selected components comprise the first $n$, and thus $q_j = p_j$ for $j \leq n$. As pointed out previously (Lerner and Parr, 2001), this approach is appropriate in many real world settings in which a large fraction of the probability mass of $P(V)$ is contained in a small number of components. This is the case for us as weather regimes tend to persist for a number of hours, and thus regime sequences with frequent regime switching are highly unlikely.

### 2.1.1 Approach 1: PRIOR

We consider three approaches to choosing the components in $Q$. Our first approach is the method of Lerner and Parr (2001) that chooses the components associated with the $n$ *a-priori* most likely state sequences, which, since our discrete variables form a simple chain, we can find efficiently using the Best Max-Marginal First (BMMF) method (Yanover and Weiss, 2003). The mixing proportions are set so that $\pi_i \propto \omega_i$. Although not theoretically justified in Lerner and Parr (2001), we show here that this choice in fact minimizes an upper bound on the *a-priori* or evidence-free KL divergence from $Q$ to $P$, $D(Q||P)$, among all $Q$ made from $n$ components of $P$. To see this we first extend $Q$ to have $N$ components where $\pi_j = 0$ for $j > n$ and apply an upper bound on the KL-divergence between mixtures of Gaussians (Singer and Warmuth, 1998; Do, 2003), $D(Q||P) \leq D(\pi||\omega) + \sum_{j=1}^{N} \pi_j D(q_j||p_j)$. Since we constrain $Q$ to have components from $P$, the second term drops out, and $D(Q||P) \leq D(\pi||\omega) = \sum_{j=1}^{n} \frac{\omega_j}{Z} \left( \log(\frac{\omega_j}{Z}) - \log(\omega_j) \right)$, where here we use that $\pi_j \propto \omega_j$ where the proportionality constant $Z = \sum_{j=1}^{n} \omega_i$ is the sum of chosen mixing probabilities. This leaves $D(Q||P) \leq -\log(Z)$, which is clearly minimized by choosing the $n$ components of $P$ with largest $\omega_j$. We call this approach $\mathsf{PRIOR}(n)$.

### 2.1.2 Approach 2: HOMO

Our second method for setting $Q(V)$ is a simple but often effective approach that assumes no regime changes in the truncated model. This approach, which we call $\mathsf{HOMO}$ has $n = C$ components, one for each homogeneous regime sequence. If the self transition probabilities $T(r, r)$ are largest, then the most likely regime sequence *a-priori* is homogeneous, and thus is also chosen by $\mathsf{PRIOR}(n)$. The other components of $\mathsf{PRIOR}(n)$ may be only small variations from this homogeneous regime sequence, however, the components selected by $\mathsf{HOMO}$ are very different from one another. This diversity may be advantageous for prediction.

### 2.1.3 Approach 3: POST

Our final method depends on the evidence. We would like to select the components for the top $n$ regime sequences with maximum *posterior* likelihood, however, this too is NP-hard (Lerner and Parr, 2001). We instead use a fast approximation in which the posterior potential of settings to regime variables is set by local evidence. We define $\tau_t(r) = \Pr(\dot{\mathbf{w}}_t | \dot{\mathbf{w}}_{t-1}, R_t = r)$ to

**Table 1:** Missing data summary. The "Count" row lists the number of hours in our WI data set (21 sites total) in which the number of sites with missing values was exactly equal to the value "# Sites Missing".

| # Sites Missing | 0 | 1 | 2 | 3 | 4 | 5 | 6 | 7+ |
|---|---|---|---|---|---|---|---|---|
| Count | 1978 | 2583 | 6463 | 8165 | 8849 | 4769 | 1229 | 1028 |
| Frequency | 5.6% | 7.4% | 18.4% | 23.3% | 25.2% | 13.6% | 3.5% | 2.9% |

be the potential for $R_t = r$, and then run BMMF on the model where $\Pr(r_{t-K}, \cdots, r_{t+K}) \propto \tau_{t-k}(r_{t-K}) \prod_{t'=t-K+1}^{t+K} \tau_{t'}(r_{t'}) T(r_{t'-1}, r_{t'})$. Note that each $\tau_t(r)$ is the density value of a single Gaussian, and can be computed quickly from model parameters. We call this approach $\mathsf{POST}(n)$.

## 3 Experimental evaluation

We compare the forecasts of our models to forecasts of persistence models, which represent the current state-of-the-art for STWF. We design our experiments to answer the following questions. 1. Are STWFs of our single-regime models more accurate than persistence forecasts? 2. Are STWFs of our models that consider regimes more accurate than those of the single-regime models? 3. Do differences between learned regimes make sense meteorologically? Additionally, we wish to comparatively evaluate the effectiveness of our approximate inference algorithm.

### 3.1 Data set

We conduct our evaluation in two meteorologically distinct regions in the United States: Wisconsin (WI), and the Pacific Northwest (PNW) states of Washington and Oregon. The National Climatic Data Center (NCDC) maintains a publicly accessible database of hourly historical climatic surface data, from which we obtained 4 years of data from a number of sites in both regions. The WI observations span from January 1, 2006 through December 31, 2009, and the PNW observations span from February 4, 2006 through February 3, 2010. We have data from 21 and 24 sites in WI and PNW, respectively. This data is available at `http://ganymede.cs.uwm.edu/nips2010/`.

We collect wind direction and wind speed at each site, as measured at 10 meters above ground level. Since our primary motivation is wind *power* forecasting, we prefer wind speed measurements taken at turbine height (approximately 50-100 meters above ground level). Publicly available turbine height observations, however, are scarce, so we use the 10 m. data as a compromise and for proof of concept.

Raw data from the NCDC is approximately hourly, but readings often appear off-the-hour in an unpredictable fashion. We use the simple rule of selecting the nearest data point within $+/-10$ minutes of the hourly transition. We discard all readings outside this margin. Additionally, NCDC appends various quality flags to each reading, and we discarded any data which did not pass all quality checks. These discarded points as well as varying site-specific instrumentation practices introduce missing observations. Table 1 shows a summary of missing data in the the WI data set. Missing data did not arise from a few misbehaving stations.

### 3.2 Experimental methodology

Data was assembled into four cross-validation folds, each contiguous and exactly 1 year in length. For each fold we use the three training years to learn $\mathsf{AR\text{-}LG}(C)$ models with $C = 1, 2, ..., 5$. Note that $\mathsf{AR\text{-}LG}(1)$ is the standard (non-conditional) auto-regressive model. With each learned model we forecast wind speeds at all sites and all test-year hours at six horizons (1-6 hours). Thus, for each geography (WI and PNW) we have 20 learned models (4 folds and $C = 1, 2, ..., 5$) and 120 prediction sequences (horizon 1-6 hrs for each of the 20 learned models). Note that this entails "casting out" or unrolling a learned model to reach longer horizons, which as we see below can impact performance. For the persistence model, we only make a horizon $h$ forecast for target time $t + h$ if the time $t$ observation at that site is available. For point-predictions we predict the expected value of the posterior wind distribution at the prediction time.

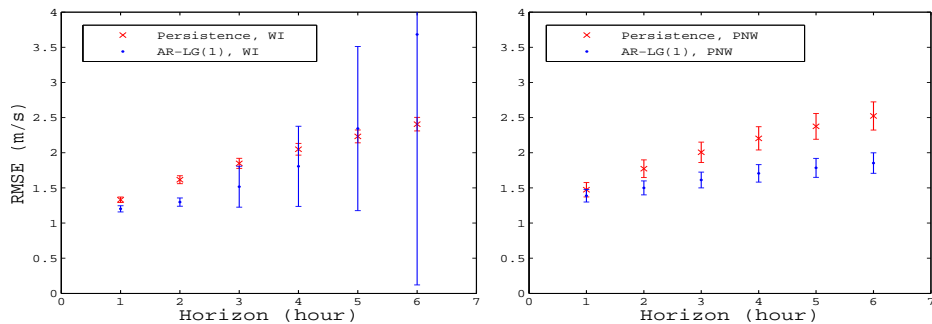

**Figure 2:** Mean RMSE over all sites and folds for single-regime model (AR-LG(1)) and persistence models in WI (left) and PNW (right). Errorbars extend one standard deviation above and below the mean.

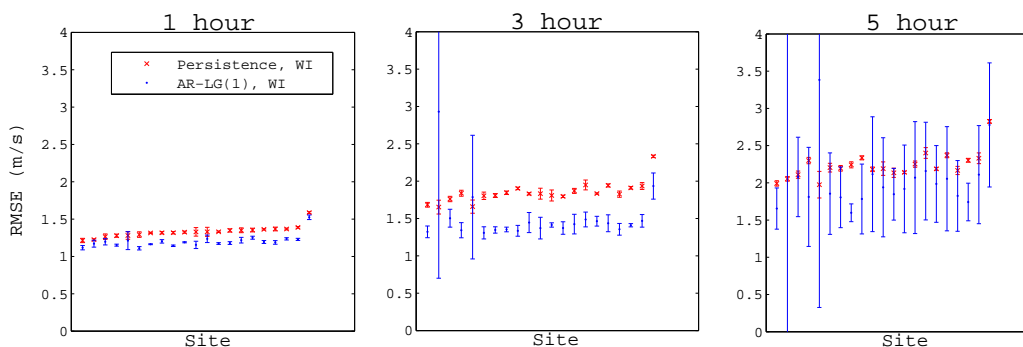

**Figure 3:** Site-by-site average RMSE values for AR-LG(1) and persistence models in WI for 1, 3 and 5 hour horizons. Errorbars show standard deviations calculated across folds (years).

We chose for these experiments to use the HOMO approximation method, with a truncated model corresponding to 3 hours ($K = 1$). This approximation method is simplest and suits our domain, where we expect distinct regimes to generally persist over a period of a few hours.

We use two performance measures to evaluate prediction sequences, test-set log-likelihood (LL) and root mean squared error (RMSE). The RMSE measure provides an evaluation for point predictions while the LL provides an evaluation of probabilistic predictions. For a given geographical region we denote the RMSE of the horizon $h$ prediction sequence made by AR-LG($C$) model for site $s$ and year $y$ by $e(h, s, y, C)$. In a similar way we denote RMSE of a persistence prediction sequence by $e_p(h, s, y)$. We denote collections and aggregates with MATLAB-style notation. For example, $e(1, :, :, 2)$ is the collection of RMSE values for 1 hour predictions for the 2 regime model across all sites and years, and $\mathsf{mean}\,[e(1, :, :, 2)]$ and $\mathsf{std}\,[e(1, :, :, 2)]$ are the collection's mean and standard deviation.

We calculate LL values of the AR-LG($C$) models relative to LL values of a persistent Gaussian model $\mathbf{w}_{t+h} = \mathbf{w}_t + \epsilon_h$. Here, $\epsilon_h$ is the horizon $h$ zero-mean Gaussian noise vector with variance estimated from the training-set.

In order to make meaningful comparisons between AR-LG($C$) and persistence models, we calculate performance measures for all horizon $h$ prediction sequences from only hours for which a corresponding horizon $h$ persistence prediction is available.

### 3.3 Results

To compare our approximation methods, we evaluate the three approximate inference procedures, HOMO, PRIOR(2) and POST(2) using simulated data from a situation with ten sites arrayed linearly (eg, east-to-west) and two regimes. The parameters of regime 1 were set for a east-to-west

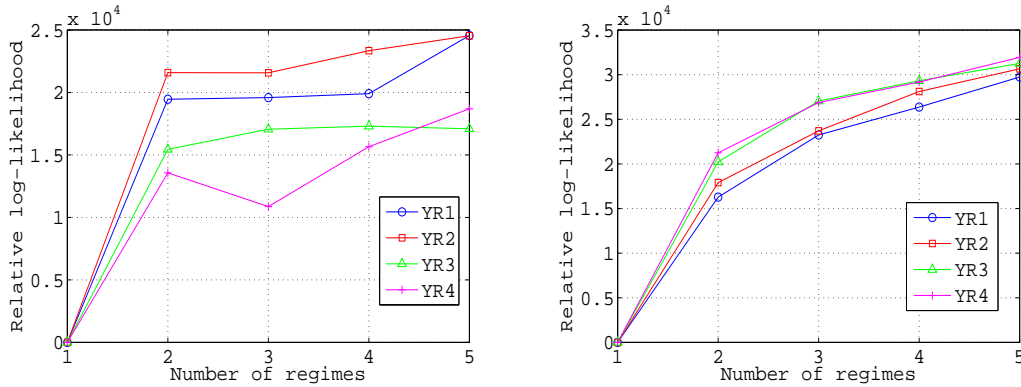

**Figure 4:** Performance of multi- versus single-regime models at a 2 hour prediction horizon, in each of 4 folds. Performance measures of test-set relative log-likelihood across all sites for WI (left) and PNW (right), with number of regimes, $C$, on the $x$-axis. Note that at $C = 1$ the value is zero since this is comparison of AR-LG(1) against itself.

moving weather regime and the parameters for regime 2 were set for a west-to-east weather regime. Self transition probabilities were set to 0.8. Observations were generated using these models and 20% were hidden, which is consistent with the missing rate in our data sets. Then we make 2-hr ahead forecasts at all time points using window size $K$ from 2 to 15 hours. The mean-absolute error of PRIOR(2) was highest for all $K$ (1.05), HOMO had the lowest overall error (0.95), and there was surprisingly no obvious trends due to $K$. The good performance of HOMO supports our hypothesis that the performance of PRIOR(2) will suffer from lack of diversity, however, we expected POST(2) to perform better relative to HOMO, but instead it had an overall error of 0.965.

In all further experiments, we attempted to assess the effectiveness of the AR-LG($C$) models, using the real wind data described above.

To answer the first question above, we compute mean $[e(h, :, :, 1)]$ and mean $[e_p(h, :, :)]$ for the RMSE collections of the AR-LG(1) and persistence models for both geographical locations and all horizons $h = 1, 2, ..., 6$. Figure 2 plots the mean RMSE of these collections. The errorbars extend 1 standard deviation unit above and below the mean. Not surprisingly, error increases with horizon length. In both WI and PNW the AR-LG(1) model has significantly lower RMSE than persistence for 1 and 2 hour time horizons. For longer horizons the results vary by geography. In PNW the gap between AR-LG(1) and persistence grows with $h$, while in in WI the AR-LG(1) performance begins to degrade relative to persistence starting with $h = 3$. At 3 and 4 hour horizons we see an increase in the variance of AR-LG(1), but still a lower mean RMSE than persistence. For $h = 5$ and $h = 6$ the persistence model has lower mean RMSE than the AR-LG(1).

To gain insight into decreasing performance at longer horizons in WI, we plot in Figure 3 the mean and standard deviation RMSE values for the site specific collections $e(h, s, :, 1)$ and $e_p(h, s, :)$ at all WI sites for 1,3 and 5 hour horizons. Each collection here contains four RMSE values, one per fold. For $h = 1$ our AR-LG(1) model beats persistence at all sites, usually by by multiple standard deviation units. This is a significant result because persistence forecasts have been shown to be difficult to improve upon for very short horizons. At $h = 3$ problems begin to appear. Although at most sites AR-LG(1) has improved further upon persistence accuracy, two sites display high variance and one (second from left) has high variance and very high RMSE near 3 m/s. At $h = 5$ high variance is widespread and the RMSE of the ill behaving sites at $h = 3$ have grown. This suggests that large erroneous predictions at a small number of sites spread throughout the system as it evolves forward in time.

Next, we consider the performance of multiple regime models. For these models we focus on the LL measure. Figure 4 plots total LL values for AR-LG(2), AR-LG(3), AR-LG(4) and AR-LG(5) relative to AR-LG(1) for individual years. In both WI and PNW there is a large jump from 1 to 2 regimes. While in WI there is no obvious trend from 2 to 5 regimes, in PNW there is a clear increase in performance as the number of regimes increases.

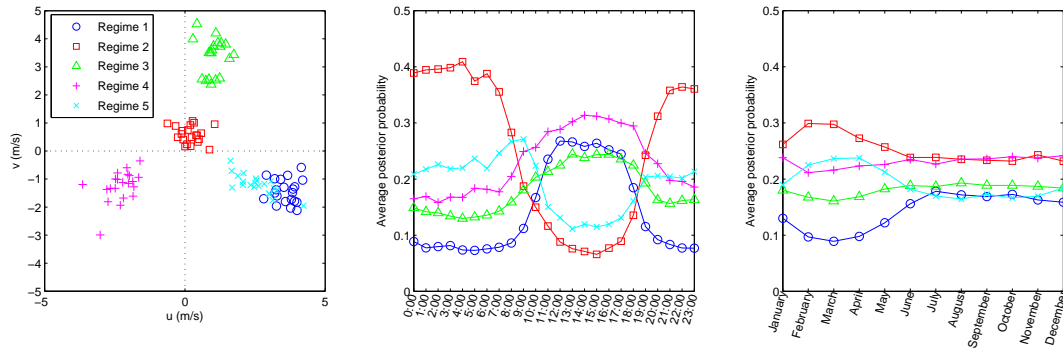

**Figure 5:** Meteorological properties of learned regimes of AR-LG(5) models in WI. (a) Mean wind vectors $(u, v)$ at each of 21 sites in WI, in each of 5 regimes (regime indicated by shape). (b) Mean regime posteriors with respect to test-set hour-of-day (CST), showing diurnal trends. (c) Mean regime posteriors with respect to test-set month.

Increase in forecast skill and test-set log-likelihood indicate that regimes in the multi-regime models are capturing important generalizable patterns in short-term wind dynamics, whose features ought to arise from underlying meteorology. Indeed, model parameters exhibit strong clustering patterns which can be tied to known regional meteorological phenomena. Figure 5 shows an analysis of a five regime model trained on the WI dataset. Figure 5 (a) plots learned wind vectors in the first time-slice between the five regimes. Figures 5 (b) and (c) analyze posterior regime likelihoods with respect to diurnal (daily) and seasonal time-frames. We note strong clusterings in (a) and significant diurnal and seasonal trends.

## 4 Conclusion

We have described a model for short-term wind forecasting (STWF), an important task in the wind power industry. Our model is set apart from previous STWF approaches in three important ways: Firstly, forecasts are informed by off-site evidence through a representation of the dynamical evolution of winds in the region. Secondly, our models can learn and reason about meteorological regimes unique to the local climate. Finally, our model is tolerant to missing data which is present in most sources of wind data. These points are shown empirically through an improvement in forecasting error versus state-of-the-art, and observation of meteorological properties of learned regimes.

We presented novel approximate inference procedures that enables AR-HMMs to be gracefully used in situations with missing data. We hope these approaches can be applied to other problem domains suited to AR-HMMs.

## Footnotes

[1]We include the additional dimension with constant value of 1.0 here to indicate that we include a constant term in all our models. But for notational simplicity in what follows we include this term only implicitly and describe our methods as if $\mathbf{W}_t$ were comprised of observations only.

## References

Caruana, R. (1997). Multitask learning. *Machine Learning*, 28:41–75.

Do, M. (2003). Fast approximation of Kullback-Leibler distance for dependence trees and hidden Markov models. *Signal Processing Letters, IEEE*, 10(4):115 – 118.

Giebel, G. (2003). The state-of-the-art in short-term prediction of wind power. Deliverable Report D1.1, Project Anemos. Available online at http://anemos.cma.fr.

Gneiting, T., Larson, K., Westrick, K., Genton, M. G., and Aldrich, E. (2006). Calibrated probabilistic forecasting at the stateline wind energy center. *Journal of the American Statistical Association*, 101(475):968–979.

Koller, D. and Friedman, N. (2009). *Probabilistic Graphical Models: Principles and Techniques*. MIT Press.

Kusiak, A., Zheng, H., and Song, Z. (2009). Short-term prediction of wind farm power: A data mining approach. *IEEE Transactions on Energy Conversion*, 24(1):125–136.

Lauritzen, S. L. and Jensen, F. (2001). Stable local computation with conditional Gaussian distributions. *Statistics and Computing*, 11:191–203.

Lerner, U. and Parr, R. (2001). Inference in hybrid networks: Theoretical limits and practical algorithms. In Breese, J. and Koller, D., editors, *Proceedings of the Seventeenth Conference on Uncertainty in Artificial Intelligence (UAI-01)*, pages 310–318, San Francisco, CA. Morgan Kaufmann Publishers.

Lindenberg, S. (2008). 20% wind energy by 2030: Increasing wind energy's contribution to U.S. electricity supply. US Department of Energy Report.

Marti, I., San Isidro, M., Cabezn, D., Loureiro, Y., Villanueva, J., Cantero, E., and Perez, I. (2004). Wind power prediction in complex terrain: from the synoptic scale to the local scale. In *EAWE Conference,The science of making torque from wind*, Delft, The Netherlands.

Murphy, K. (1998). Fitting a conditional linear Gaussian distribution. http://www.cs.ubc.ca/ murphyk/Papers/learncg.pdf.

Paroli, R., Pistollato, S., Rosa, M., and Spezia, L. (2005). Non-homogeneous markov mixture of periodic autoregressions for the analysis of air pollution in the lagoon of venice. In *Applied Stochastic Models and Data Analysis (ASMDA-2005)*, pages 1124–1132.

Perez-Quiros, G., Camacho, M., and Poncela, P. (2010). Green shoots? Where, when and how? Working Papers 2010-04, FEDEA.

Pinson, P. and Madsen, H. (2008). Probabilistic forecasting of wind power at the minute time-scale with markov-switching autoregressive models. *Imagine*.

Piwko, D. and Jordan, G. (2010). The economic value of day-ahead wind forecasts for power grid operations. 2010 UWIG Workshop on Wind Forecasting.

Singer, Y. and Warmuth, M. K. (1998). Batch and on-line parameter estimation of Gaussian mixtures based on the joint entropy. In Kearns, M. J., Solla, S. A., and Cohn, D. A., editors, *NIPS*, pages 578–584. The MIT Press.

Tang, X. (2004). Autoregressive hidden markov model with application in an El Niño study. Master's thesis, University of Saskatchewan, Saskatoon, Saskatchewan, Canada.

Toscani, D., Archetti, F., Quarenghi, L., Bargna, F., and Messina, E. (2010). A DSS for assessing the impact of environmental quality on emergency hospital admissions. In *Health Care Management (WHCM), 2010 IEEE Workshop on*, pages 1 –6.

Weiss, Y. and Freeman, W. T. (2001). Correctness of belief propagation in Gaussian graphical models of arbitrary topology. *Neural Computation*, 13(10):2173–2200.

Yanover, C. and Weiss, Y. (2003). Finding the M most probable configurations using loopy belief propagation. In Thrun, S., Saul, L. K., and Schölkopf, B., editors, *NIPS*. MIT Press.

Zheng, H. and Kusiak, A. (2009). Prediction of wind farm power ramp rates: A data-mining approach. *Journal of Solar Energy Engineering*.

